# Hierarchical Optimistic Region Selection driven by Curiosity

**Odalric-Ambrym Maillard**
Lehrstuhl für Informationstechnologie
Montanuniversität Leoben
Leoben, A-8700, Austria
`odalricambrym.maillard@gmail.com`

## Abstract

This paper aims to take a step forwards making the term "intrinsic motivation" from reinforcement learning theoretically well founded, focusing on curiosity-driven learning. To that end, we consider the setting where, a fixed partition $\mathcal{P}$ of a continuous space $\mathcal{X}$ being given, and a process $\nu$ defined on $\mathcal{X}$ being unknown, we are asked to sequentially decide which cell of the partition to select as well as where to sample $\nu$ in that cell, in order to minimize a loss function that is inspired from previous work on curiosity-driven learning. The loss on each cell consists of one term measuring a simple worst case quadratic sampling error, and a penalty term proportional to the range of the variance in that cell. The corresponding problem formulation extends the setting known as active learning for multi-armed bandits to the case when each arm is a continuous region, and we show how an adaptation of recent algorithms for that problem and of hierarchical optimistic sampling algorithms for optimization can be used in order to solve this problem. The resulting procedure, called Hierarchical Optimistic Region SElection driven by Curiosity (HORSE.C) is provided together with a finite-time regret analysis.

## 1 Introduction

In this paper, we focus on the setting of intrinsically motivated reinforcement learning (see Oudeyer and Kaplan [2007], Baranes and Oudeyer [2009], Schmidhuber [2010], Graziano et al. [2011]), which is an important emergent topic that proposes new difficult and interesting challenges for the theorist. Indeed, if some formal objective criteria have been proposed to implement specific notions of intrinsic rewards (see Jung et al. [2011], Martius et al. [2007]), so far, many - and only - experimental work has been carried out for this problem, often with interesting output (see Graziano et al. [2011], Mugan [2010], Konidaris [2011]) but unfortunately no performance guarantee validating a proposed approach. Thus proposing such an analysis may have great immediate consequences for validating some experimental studies.

**Motivation.** A typical example is the work of Baranes and Oudeyer [2009] about curiosity-driven learning (and later on Graziano et al. [2011], Mugan [2010], Konidaris [2011]), where a precise algorithm is defined together with an experimental study, yet no formal goal is defined, and no analysis is performed as well. They consider a so-called sensory-motor space $\mathcal{X} \stackrel{\text{def}}{=} \mathcal{S} \times \mathcal{M} \subset [0,1]^d$ where $\mathcal{S}$ is a (continuous) state space and $\mathcal{M}$ is a (continuous) action space. There is no reward, yet one can consider that the goal is to actively select and sample subregions of $\mathcal{X}$ for which a notion of "learning progress" - this intuitively measures the decay of some notion of error when successively sampling into one subregion - is maximal. Two key components are advocated in Baranes and Oudeyer [2009], in order to achieve successful results (despite that success is a fuzzy notion):

- The use of a *hierarchy* of regions, where each region is progressively split into sub-regions.

- Splitting leaf-regions in two based on the optimization of the dissimilarity, amongst the regions, of the learning progress. The idea is to identify regions with a learning complexity that is a globally constant in that region, which also provides better justification for allocating samples between identified regions.

We believe it is possible to go one step towards a full performance analysis of such algorithms, by relating the corresponding active sampling problem to existing frameworks.

**Contribution.** This paper aims to take a step forwards making the term "intrinsic motivation" from reinforcement learning theoretically well founded, focusing on curiosity-driven learning. We introduce a mathematical framework in which a metric space (which intuitively plays the role of the state-action space) is divided into regions and a learner has to sample from an unknown random function in a way that reduces a notion of error measure the most. This error consists of two terms, the first one is a robust measure of the quadratic error between the observed samples and their unknown mean, the second one penalizes regions with non constant learning complexity, thus enforcing the notion of curiosity. The paper focuses on how to choose the region to sample from, when a partition of the space is provided.

The resulting problem formulation can be seen as a non trivial extension of the setting of active learning in multi-armed bandits (see Carpentier et al. [2011] or Antos et al. [2010]), where the main idea is to estimate the variance of each arm and sample proportionally to it, to the case when each arm is a region as opposed to a point. In order to deal with this difficulty, the maximal and minimal variance inside each region is tracked by means of a hierarchical optimization procedure, in the spirit of the HOO algorithm from Bubeck et al. [2011]. This leads to a new procedure called Hierarchical Optimistic Region SElection driven by Curiosity (HORSE.C) for which we provide a theoretical performance analysis.

**Outline.** The outline of the paper is the following. In Section 2 we introduce the precise setting and define the objective function. Section 3 defines our assumptions. Then in Section 4 we present the HORSE.C algorithm. Finally in Section 5, we provide the main Theorem 1 that gives performance guarantees for the proposed algorithm.

## 2 Setting: Robust region allocation with curiosity-inducing penalty.

Let $\mathcal{X}$ assumed to be a metric space and let $\mathcal{Y} \subset \mathbb{R}^d$ be a normed space, equipped with the Euclidean norm $|| \cdot ||$. We consider an unknown $\mathcal{Y}$-valued process defined on $\mathcal{X}$, written $\nu : \mathcal{X} \to \mathfrak{M}_1^+(\mathcal{Y})$, where $\mathfrak{M}_1^+(\mathcal{Y})$ refers to the set of all probability measures on $\mathcal{Y}$, such that for all $x \in \mathcal{X}$, the random variable $Y \sim \nu(x)$ has mean $\mu(x) \in \mathbb{R}^d$ and covariance matrix $\Sigma(x) \in \mathcal{M}_{d,d}(\mathbb{R})$ assumed to be diagonal. We thus introduce for convenience the notation $\rho(x) \stackrel{\text{def}}{=} trace(\Sigma(x))$, where $trace$ is the trace operator (this corresponds to the variance in dimension 1). We call $\mathcal{X}$ the input space or sampling space, and $\mathcal{Y}$ the output space or value space.

**Intuition** Intuitively when applied to the setting of Baranes and Oudeyer [2009], then $\mathcal{X} \stackrel{\text{def}}{=} \mathcal{S} \times \mathcal{A}$ is the space of state-action pairs, where $\mathcal{S}$ is a continuous state space and $\mathcal{A}$ a continuous action space, $\nu$ is the transition kernel of an unknown MDP, and finally $\mathcal{Y} \stackrel{\text{def}}{=} \mathcal{S}$. This is the reason why we consider $\mathcal{Y} \subset \mathbb{R}^d$ and not only $\mathcal{Y} \subset \mathbb{R}$ as would seem more natural. One difference is that we assume (see Section 3) that we can sample anywhere in $\mathcal{X}$, which is a restrictive yet common assumption in the reinforcement learning literature. How to get rid of this assumption is an open and challenging question that is left for future work.

**Sampling error and robustness** Let us consider a sequential sampling process on $\mathcal{X}$, i.e. a process that samples at time $t$ a value $Y_t \sim \nu(X_t)$ at point $X_t$, where $X_t \in \mathcal{F}_{<t}$ is a measurable function of the past inputs and outputs $\{(X_s, Y_s)\}_{s<t}$. It is natural to look at the following quantity, that we call average noise vector $\eta_t$:

$$\eta_t \stackrel{\text{def}}{=} \frac{1}{t} \sum_{s=1}^{t} Y_s - \mu(X_s) \in \mathbb{R}^d .$$

One interesting property is that this is a martingale difference sequence, which means that the norm of this vector enjoys a concentration property. More precisely (see [Maillard, 2012, Lemma 1] in the extended version of the paper), we have for all deterministic $t > 0$

$$\mathbb{E}[ ||\eta_t||^2 ] = \frac{1}{t}\mathbb{E}\Big[\frac{1}{t} \sum_{s=1}^{t} \rho(X_s)\Big] .$$

A similar property holds for a region $R \subset \mathcal{X}$ that has been sampled $n_t(R)$ times, and in order to be robust against a bad sampling strategy inside a region, it is natural to look at the worst case error, that we define as

$$\mathbf{e}_R(n_t) \stackrel{\text{def}}{=} \frac{\sup_{x \in R} \rho(x)}{n_t(R)}.$$

One reason for looking at robustness is that for instance, in the case we work with an MDP, we are generally not completely free to choose the sample $X_s \in \mathcal{S} \times \mathcal{A}$: we can only choose the action and the next state is generally given by Nature. Thus, it is important to be able to estimate this worst case error so as to prevent from bad situations.

**Goal** Now let $\mathcal{P}$ be a fixed, known partition of the space $\mathcal{X}$ and consider the following game. The goal of an algorithm is, at each time step $t$, to propose one point $x_t$ where to sample the space $\mathcal{X}$, so that its allocation of samples $\{n_t(R)\}_{R \in \mathcal{P}}$ (that is, the number of points sampled in each region) minimizes some objective function. Thus, the algorithm is free to sample everywhere in each region, with the goal that the total number of points chosen in each region is optimal in some sense. A simple candidate for this objective function would be the following

$$L_{\mathcal{P}}(n_t) \stackrel{\text{def}}{=} \max \left\{ \mathbf{e}_R(n_t) ;\ R \in \mathcal{P} \right\},$$

however, in order to incorporate a notion of curiosity, we would also like to penalize regions that have a variance term $\rho$ that is non homogeneous (i.e. the less homogeneous, the more samples we allocate). Indeed, if a region has constant variance, then we do not really need to understand more its internal structure, and thus its better to focus on an other region that has very heterogeneous variance. For instance, one would like to split such a region in several homogeneous parts, which is essentially the idea behind section C.3 of Baranes and Oudeyer [2009]. We thus add a curiosity-penalization term to the previous objective function, which leads us to define the pseudo-loss of an allocation $n_t \stackrel{\text{def}}{=} \{n_t(R)\}_{R \in \mathcal{P}}$ in the following way:

$$\mathfrak{L}_{\mathcal{P}}(n_t) \stackrel{\text{def}}{=} \max \left\{ \mathbf{e}_R(n_t) + \lambda |R| (\max_{x \in R} \rho(x) - \min_{x \in R} \rho(x)) ;\ R \in \mathcal{P} \right\}. \tag{1}$$

Indeed, this means that we do not want to focus just on regions with high variance, but also trade-off with highly heterogeneous regions, which is coherent with the notion of curiosity (see Oudeyer and Kaplan [2007]). For convenience, we also define the pseudo-loss of a region $R$ by

$$\mathcal{L}_R(n_t) \stackrel{\text{def}}{=} \mathbf{e}_R(n_t) + \lambda |R| (\max_{x \in R} \rho(x) - \min_{x \in R} \rho(x)).$$

**Regret** The regret (or loss) of an allocation algorithm at time $T$ is defined as the difference between the cumulated pseudo-loss of the allocations $n_t = \{n_{R,t}\}_{R \in \mathcal{P}}$ proposed by the algorithm and that of the best allocation strategy $n_t^\star = \{n_{R,t}^\star\}_{R \in \mathcal{P}}$ at each time steps; we define

$$\mathfrak{R}_T \stackrel{\text{def}}{=} \sum_{t=|\mathcal{P}|}^{T} \mathfrak{L}_{\mathcal{P}}(n_t) - \mathfrak{L}_{\mathcal{P}}(n_t^\star),$$

where an optimal allocation at time $t$ is defined by

$$n_t^\star \in \operatorname{argmin} \left\{ \mathfrak{L}_{\mathcal{P}}(n_t) ;\ \{n_t(R)\}_{R \in \mathcal{P}} \text{ is such that } \sum_{R \in \mathcal{P}} n_t(R) = t \right\}.$$

Note that the sum starts at $t = |\mathcal{P}|$ for a technical reason, since for $t < |\mathcal{P}|$, whatever the allocation, there is always at least one region with no sample, and thus $\mathfrak{L}_{\mathcal{P}}(n_t) = \infty$.

**Example 1** In the special case when $\mathcal{X} = \{1, \ldots, K\}$ is finite with $K \ll T$, and when $\mathcal{P}$ is the complete partition (each cell corresponds to exactly one point), the penalization term is canceled. Thus the problem reduces to the choice of the quantities $n_t(i)$ for each arm $i$, and the loss of an allocation simply becomes

$$\mathfrak{L}(n_t) \stackrel{\text{def}}{=} \max \left\{ \frac{\rho(i)}{n_t(i)} ;\ 1 \leq i \leq K \right\}.$$

This almost corresponds to the already challenging setting analyzed for instance in Carpentier et al. [2011] or Antos et al. [2010]. The difference is that we are interested in the *cumulative* regret of our allocation instead of only the regret suffered for the last round as considered in Carpentier et al. [2011] or Antos et al. [2010]. Also we directly target $\frac{\rho(i)}{n_t(i)}$ whereas they consider the mean sampling error (but both terms are actually of the same order). Thus the setting we consider can be seen as a generalization of these works to the case when each arm corresponds to a continuous sampling domain.

# 3 Assumptions

In this section, we introduce some mild assumptions. We essentially assume that the unknown distribution is such that it has a sub-Gaussian noise, and a smooth mean and variance functions. These are actually very mild assumptions. Concerning the algorithm, we consider it can use a partition tree of the space, and that this one is essentially not degenerated (a typical binary tree that satisfies all the following assumptions is such that each cell is split in two children of equal volume). Such assumptions on trees have been extensively discussed for instance in Bubeck et al. [2011].

**Sampling** At any time, we assume that we are able to sample at any point in $\mathcal{X}$, i.e. we assume we have a generative model[1] of the unknown distribution $\nu$.

**Unknown distribution** We assume that $\nu$ is sub-Gaussian, meaning that for all fixed $x \in \mathcal{X}$

$$\forall \lambda \in \mathbb{R}^d \ \ln \mathbb{E} \exp[\langle \lambda, Y - \mu(X) \rangle] \leq \frac{\lambda^T \Sigma(x) \lambda}{2} \,,$$

and has diagonal covariance matrix in each point[2].

The function $\mu$ is assumed to be Lipschitz w.r.t a metric $\ell_1$, i.e. it satisfies

$$\forall x, x' \in \mathcal{X} \ ||\mu(x) - \mu(x')|| \leq \ell_1(x, x') \,.$$

Similarly, the function $\rho$ is assumed to be Lipschitz w.r.t a metric $\ell_2$, i.e. it satisfies

$$\forall x, x' \in \mathcal{X} \ |\rho(x) - \rho(x')| \leq \ell_2(x, x') \,.$$

**Hierarchy** We assume that $\mathcal{Y}$ is a convex and compact subset of $[0, 1]^d$. We consider an infinite binary tree $\mathcal{T}$ whose nodes correspond to regions of $\mathcal{X}$. A node is indexed by a pair $(h, i)$, where $h \geq 0$ is the depth of the nodes in $\mathcal{T}$ and $0 \leq i < 2^h$ is the position of the node at depth $h$. We write $R(h, i) \subset \mathcal{X}$ the region associated with node $(h, i)$. The regions are fixed in advance, are all assumed to be measurable with positive measure, and must satisfy that for each $h \geq 1$, $\{R(h, i)\}_{0 \leq i < 2^h}$ is a partition of $\mathcal{X}$ that is compatible with depth $h - 1$, where $R(0, 0) \stackrel{\text{def}}{=} \mathcal{X}$; in particular for all $h \geq 0$, for all $0 \leq i < 2^h$, then

$$R(h, i) = R(h + 1, 2i) \cup R(h + 1, 2i + 1) \,.$$

In dimension $d$, a standard way to define such a tree is to split each parent node in half along the largest side of the corresponding hyper-rectangle, see Bubeck et al. [2011] for details.

For a finite sub-tree $\mathcal{T}_t$ of $\mathcal{T}$, we write $Leaf(\mathcal{T}_t)$ for the set of all leaves of $\mathcal{T}_t$. For a region $(h, i) \in \mathcal{T}_t$, we denote by $\mathcal{C}_t(h, i)$ the set of its children in $\mathcal{T}_t$, and by $\mathcal{T}_t(h, i)$ the subtree of $\mathcal{T}_t$ starting with root node $(h, i)$.

**Algorithm and partition** The partition $\mathcal{P}$ is assumed to be such that each of its regions $R$ corresponds to one region $R(h, i) \in \mathcal{T}$; equivalently, there exists a finite sub-tree $\mathcal{T}_0 \subset \mathcal{T}$ such that $Leaf(\mathcal{T}_0) = \mathcal{P}$. An algorithm is only allowed to expand one node of $\mathcal{T}_t$ at each time step $t$. In the sequel, we write indifferently $\mathcal{P} \in \mathcal{T}$ and $(h, i) \in \mathcal{T}$ or $\mathcal{P}$ and $R(h, i) \subset \mathcal{X}$ to refer to the partition or one of its cell.

**Exponential decays** Finally, we assume that the $\ell_1$ and $\ell_2$ diameters of the region $R(h, i)$ as well as its volume $|R(h, i)|$ decay at exponential rate in the sense that there exists positive constants $\gamma, \gamma_1, \gamma_2$ and $c, c_1, c_2$ such that for all $h \geq 0$, then $|R(h, i)| \leq c\gamma^h$,

$$\max_{x', x \in R(h,i)} \ell_1(x, x') \leq c_1 \gamma_1^h \text{ and } \max_{x', x \in R(h,i)} \ell_2(x, x') \leq c_2 \gamma_2^h.$$

Similarly, we assume that there exists positive constants $c' \leq c$, $c_1' \leq c_1$ and $c_2' \leq c_2$ such that for all $h \geq 0$, then $|R(h, i)| \geq c'\gamma^h$,

$$\max_{x', x \in R(h,i)} \ell_1(x, x') \geq c_1' \gamma_1^h \text{ and } \max_{x', x \in R(h,i)} \ell_2(x, x') \geq c_2' \gamma_2^h.$$

This assumption is made to avoid degenerate trees and for general purpose only. It actually holds for any reasonable binary tree.

# 4 Allocation algorithm

In this section, we now introduce the main algorithm of this paper in order to solve the problem considered in Section 2. It is called Hierarchical Optimistic Region SElection driven by Curiosity. Before proceeding, we need to define some quantities.

## 4.1 High-probability upper-bound and lower-bound estimations

Let us consider the following (biased) estimator

$$\hat{\boldsymbol{\sigma}}_t^2(R) \stackrel{\text{def}}{=} \frac{1}{N_t(R)} \sum_{s=1}^t ||Y_s||^2 \mathbb{I}\{X_s \in R\} - ||\frac{1}{N_t(R)} \sum_{s=1}^t Y_s \mathbb{I}\{X_s \in R\}||^2 .$$

Apart from a small multiplicative biased by a factor $\frac{N_t(R)-1}{N_t(R)}$, it has more importantly a positive bias due to the fact that the random variables do not share the same mean; this phenomenon is the same as the estimation of the average variance for independent but non i.i.d variables with different means $\{\mu_i\}_{i\leq n}$, where the bias would be given by $\frac{1}{n} \sum_{i=1}^n [\mu_i - \frac{1}{n} \sum_{j=1}^n \mu_j]^2$ (see Lemma 5). In our case, it is thus always non negative, and under the assumption that $\mu$ is Lipschitz w.r.t the metric $\ell_1$, it is fortunately bounded by $d_1(R)^2$, the diameter of $R$ w.r.t the metric $\ell_1$.

We then introduce the two following key quantities, defined for all $x \in R$ and $\delta \in [0, 1]$ by

$$U_t(R, x, \delta) \stackrel{\text{def}}{=} \hat{\sigma}_t^2(R) + (1 + 2\sqrt{d}) \sqrt{\frac{d \ln(2d/\delta)}{2N_t(R)}} + \frac{d \ln(2d/\delta)}{2N_t(R)} + \frac{1}{N_t(R)} \sum_{s=1}^t \ell_2(X_s, x) \mathbb{I}\{X_s \in R\},$$

$$L_t(R, x, \delta) \stackrel{\text{def}}{=} \hat{\sigma}_t^2(R) - (1 + 2\sqrt{d}) \sqrt{\frac{d \ln(2d/\delta)}{2N_t(R)}} - d_1(R)^2 - \frac{1}{N_t(R)} \sum_{s=1}^t \ell_2(X_s, x) \mathbb{I}\{X_s \in R\} .$$

Note that we would have preferred to replace the terms involving $\ln(2d/\delta)$ with a term depending on the empirical variance, in the spirit of Carpentier et al. [2011] or Antos et al. [2010]. However, contrary to the estimation of the mean, extending the standard results valid for i.i.d data to the case of a *martingale difference sequence* is non trivial for the estimation of the variance, especially due to the additive bias resulting from the fact that the variables may not share the same mean, but also to the absence of such results for U-statistics (up to the author's knowledge). For that reason such an extension is left for future work.

The following results (we provide the proof in [Maillard, 2012, Appendix A.3]) show that $U_t(R, x, \delta)$ is a high probability upper bound on $\rho(x)$ while $L_t(R, x, \delta)$ is a high probability lower bound on $\rho(x)$.

**Proposition 1** *Under the assumptions that $\mathcal{Y}$ is a convex subset of $[0, 1]^d$, $\nu$ is sub-Gaussian, $\rho$ is Lipschitz w.r.t. $\ell_2$ and $R \subset \mathcal{X}$ is compact and convex, then*

$$\mathbb{P}\Big(\forall x \in \mathcal{X} \,;\, U_t(R, x, \delta) \leq \rho(x)\Big) \leq t\delta .$$

*Similarly, under the same assumptions, then*

$$\mathbb{P}\Big(\forall x \in \mathcal{X} \,;\, L_t(R, x, \delta) \leq \rho(x) - b(x, R, N_t(R), \delta)\Big) \leq t\delta ,$$

*where we introduced for convenience the quantity*

$$b(x, R, n, \delta) \quad \stackrel{\text{def}}{=} \quad 2 \max_{x' \in R} \ell_2(x, x') + d_1(R)^2 + 2(1 + 2\sqrt{d}) \sqrt{\frac{d \ln(2d/\delta)}{2n}} + \frac{d \ln(2d/\delta)}{2n} .$$

Now on the other other hand, we have that (see the proof in [Maillard, 2012, Appendix A.3])

**Proposition 2** *Under the assumptions that $\mathcal{Y}$ is a convex subset of $[0, 1]^d$, $\nu$ is sub-Gaussian, $\mu$ is Lipschitz w.r.t. $\ell_1$, $\rho$ is Lipschitz w.r.t. $\ell_2$ and $R \subset \mathcal{X}$ is compact and convex, then*

$$\mathbb{P}\Big(\forall x \in \mathcal{X} \,;\, U_t(R, x, \delta) \geq \rho(x) + b(x, R, N_t(R), \delta)\Big) \leq t\delta .$$

*Similarly, under the same assumptions, then*

$$\mathbb{P}\Big(\forall x \in \mathcal{X} \,;\, L_t(R, x, \delta) \geq \rho(x)\Big) \leq t\delta .$$

## 4.2 Hierarchical Optimistic Region SElection driven by Curiosity (HORSE.C).

The pseudo-code of the HORSE.C algorithm is presented in Figure 1 below. This algorithm relies on the estimation of the quantities $\max_{x \in R} \rho(x)$ and $\min_{x \in R} \rho(x)$ in order to define which point $X_{t+1}$ to sample at time $t+1$. It is chosen by expanding a leaf of a hierarchical tree $\mathcal{T}_t \subset \mathcal{T}$, in an optimistic way, starting with a tree $\mathcal{T}_0$ with leaves corresponding to the partition $\mathcal{P}$.

The intuition is the following: let us consider a node $(h, i)$ of the tree $\mathcal{T}_t$ expanded by the algorithm at time $t$. The maximum value of $\rho$ in $R(h, i)$ is thus achieved for one of its children node $(h', i') \in \mathcal{C}_t(h, i)$. Thus if we have computed an upper bound on the maximal value of $\rho$ in each child, then we have an upper bound on the maximum value of $\rho$ in $R(h, i)$. Proceeding in a similar way for the lower bound, this motivates the following two recursive definitions:

$$\hat{\boldsymbol{\rho}}_t^+(h, i; \delta) \stackrel{\text{def}}{=} \min\left\{ \max_{x \in R(h,i)} U_t(R(h, i), x, \delta), \max\left\{ \hat{\boldsymbol{\rho}}_t^+(h', i'; \delta); (h', i') \in \mathcal{C}_t(h, i) \right\} \right\},$$

$$\hat{\boldsymbol{\rho}}_t^-(h, i; \delta) \stackrel{\text{def}}{=} \max\left\{ \min_{x \in R(h,i)} L_t(R(h, i), x, \delta), \min\left\{ \hat{\boldsymbol{\rho}}_t^-(h', i'; \delta); (h', i') \in \mathcal{C}_t(h, i) \right\} \right\}.$$

These values are used in order to build an optimistic estimate of the quantity $\mathcal{L}_{R(h,i)}(N_t)$ in region $(h, i)$ (step 4), and then to select in which cell of the partition we should sample (step 5). Then the algorithm chooses where to sample in the selected region so as to improve the estimations of $\hat{\rho}_t^+$ and $\hat{\rho}_t^-$. This is done by alternating (step 6.) between expanding a leaf following a path that is optimistic according to $\hat{\rho}_t^+$ (step 7,8,9), or according to $\hat{\rho}_t^-$ (step 11.)

Thus, at a high level, the algorithm performs on each cell $(h, i) \in \mathcal{P}$ of the given partition two hierarchical searches, one for the maximum value of $\rho$ in region $R(h, i)$ and one for its minimal value. This can be seen as an adaptation of the algorithm HOO from Bubeck et al. [2011] with the main difference that we target the variance and not just the mean (this is more difficult). On the other hand, there is a strong link between step 5, where we decide to allocate samples between regions $\{R(h, i)\}_{(h,i) \in \mathcal{P}}$, and the CH-AS algorithm from Carpentier et al. [2011].

## 5  Performance analysis of the HORSE.C algorithm

In this section, we are now ready to provide the main theorem of this paper, i.e. a regret bound on the performance of the HORSE.C algorithm, which is the main contribution of this work. To this end, we make use of the notion of near-optimality dimension, introduced in Bubeck et al. [2011], and that measures a notion of intrinsic dimension of the maximization problem.

**Definition (Near optimality dimension)** For $c > 0$, the $c$-optimality dimension of $\rho$ restricted to the region $R$ with respect to the pseudo-metric $\ell_2$ is defined as

$$\max\left\{ \limsup_{\epsilon \to 0} \frac{\ln(\mathcal{N}(R_{c\epsilon}, \ell_2, \epsilon))}{\ln(\epsilon^{-1})}, 0 \right\} \text{ where } R_{c\epsilon} \stackrel{\text{def}}{=} \left\{ x \in R; \rho(x) \geq \max_{x \in R} \rho(x) - \epsilon \right\},$$

and where $\mathcal{N}(R_{c\epsilon}, \ell_2, \epsilon)$ is the $\epsilon$-packing number of the region $R_{c\epsilon}$.

Let $d^+(h_0, i_0)$ be the *c-optimality dimension* of $\rho$ restricted to the region $R(h_0, i_0)$ (see e.g. Bubeck et al. [2011]), with the constant $c \stackrel{\text{def}}{=} 4(2c_2 + c_1^2)/c_2'$. Similarly, let $d^-(h_0, i_0)$ be the $c$-optimality dimension of $-\rho$ restricted to the region $R(h_0, i_0)$. Let us finally define the biggest near-optimality dimension of $\rho$ over each cell of the partition $\mathcal{P}$ to be

$$d_\rho \stackrel{\text{def}}{=} \max\left\{ \max\left\{ d^+(h_0, i_0), d^-(h_0, i_0) \right\}; (h_0, i_0) \in \mathcal{P} \right\}.$$

**Theorem 1 (Regret bound for HORSE.C)** *Under the assumptions of Section 3 and if moreover $\gamma_1^2 \leq \gamma_2$, then for all $\delta \in [0, 1]$, the regret of the Hierarchical Optimistic Region SElection driven by Curiosity procedure parameterized with $\delta$ is bounded with probability higher than $1 - 2\delta$ as follows.*

$$\mathfrak{R}_T \leq \sum_{t=|\mathcal{P}|}^{T} \max_{(h_0, i_0) \in \mathcal{P}} \left( \frac{1}{n_t^\star(h_0, i_0)} + 2\lambda c \gamma^{h_0} \right) B\big(h_0, n_t^\star(h_0, i_0), \delta_t\big),$$

**Algorithm 1** The HORSE.C algorithm.

---

**Require:** An infinite binary tree $\mathcal{T}$, a partition $\mathcal{P} \subset \mathcal{T}$, $\delta \in [0,1]$, $\lambda \geq 0$

1: Let $\mathcal{T}_0$ be such that $Leaf(\mathcal{T}_0) = \mathcal{P}$, and $\delta_{i,t} = \frac{6\delta}{\pi^2 i^2 (2t+1)|\mathcal{P}|t^3}$, $t := 0$.

2: **while true do**

3:    define for each region $(h,i) \in \mathcal{T}_t$ the estimated loss

$$\hat{\mathcal{L}}_t(h,i) \stackrel{\text{def}}{=} \frac{\hat{\boldsymbol{\rho}}_t^+(h,i;\delta)}{N_t(R(h,i))} + \lambda |R(h,i)| \left(\hat{\boldsymbol{\rho}}_t^+(h,i;\delta) - \hat{\boldsymbol{\rho}}_t^-(h,i;\delta)\right),$$

where $\delta = \delta_{N_t(R(h,i)),t}$, where by convention $\hat{\mathcal{L}}_t(h,i)$ if it is undefined.

4:    choose the next region of the current partition $\mathcal{P} \subset \mathcal{T}$ to sample

$$(H_{t+1}, I_{t+1}) \stackrel{\text{def}}{=} \text{argmax}\left\{\hat{\mathcal{L}}_t(h,i) ; (h,i) \subset \mathcal{P}\right\}.$$

5:    **if** $N_t(R(h,i)) = n$ is odd **then**

6:       sequentially select a path of children of $(H_{t+1}, I_{t+1})$ in $\mathcal{T}_t$ defined by the initial node $(H_{t+1}^0, I_{t+1}^0) \stackrel{\text{def}}{=} (H_{t+1}, I_{t+1})$, and then

$$(H_{t+1}^{j+1}, I_{t+1}^{j+1}) \stackrel{\text{def}}{=} \text{argmax}\left\{\hat{\boldsymbol{\rho}}_t^+(h,i;\delta_{n,t}) ; (h,i) \in \mathcal{C}_t(H_{t+1}^j, I_{t+1}^j)\right\},$$

until $j = j_{t+1}$ is such that $(H_{t+1}^{j_{t+1}}, I_{t+1}^{j_{t+1}}) \in Leaf(\mathcal{T}_t)$.

7:       expand the node $(H_{t+1}^{j_{t+1}}, I_{t+1}^{j_{t+1}})$ in order to define $\mathcal{T}_{t+1}$ and then define the candidate child

$$(h_{t+1}, i_{t+1}) \stackrel{\text{def}}{=} \text{argmax}\left\{\hat{\boldsymbol{\rho}}_t^+(h,i;\delta_{n,t}) ; (h,i) \in \mathcal{C}_{t+1}(H_{t+1}^{j_{t+1}}, I_{t+1}^{j_{t+1}})\right\}.$$

8:       sample at point $X_{t+1}$ and receive the value $Y_{t+1} \sim \nu(X_{t+1})$, where

$$X_{t+1} \stackrel{\text{def}}{=} \text{argmax}\left\{U_t(R(h_{t+1}, i_{t+1}), x, \delta_{n,t}) ; x \in R(h_{t+1}, i_{t+1})\right\},$$

9:    **else**

10:      proceed similarly than steps 6,7,8 with $\hat{\boldsymbol{\rho}}_t^+$ replaced with $\hat{\boldsymbol{\rho}}_t^-$.

11:    **end if**

12:    $t := t + 1$.

13: **end while**

---

where $\delta_t$ is a shorthand notation for the quantity $\delta_{n_t^\star(h_0,i_0),t-1}$, where $n_t^\star(h_0,i_0)$ is the optimal allocation at round $t$ for the region $(h_0,i_0) \in \mathcal{P}$ and where

$$B(h_0, k, \delta_{k,t}) \stackrel{\text{def}}{=} \min_{h_0 \leq h} \left\{ 2c_2\gamma_2^h + c_1^2\gamma_1^{2h} + 2(1+2\sqrt{d})\sqrt{\frac{d\ln(2d/\delta_{k,t})}{2N_{h_0}(h,k)}} + \frac{d\ln(2d/\delta_{k,t})}{2N_{h_0}(h,k)} \right\},$$

in which we have used the following quantity

$$N_{h_0}(h,k) \stackrel{\text{def}}{=} \frac{1}{C(c_2'\gamma_2^h)^{-d_\rho}}\left(k - 2^{h-h_0}[2 + 4\sqrt{d} + \sqrt{d\ln(2d/\delta_{k,t})/2}]^2 \frac{d\ln(2d/\delta_{k,t})}{2(2c_2\gamma_2^h + c_1^2\gamma_1^{2h})^2}\right).$$

Note that the assumption $\gamma_1^2 \leq \gamma_2$ is only here so that $d_\rho$ can be defined w.r.t the metric $\ell_2$ only. We can remove it at the price of using instead a metric mixing $\ell_1$ and $\ell_2$ together and of much more technical considerations. Similarly, we could have expressed the result using the local values $d^+(h_0, i_0)$ instead of the less precise $d_\rho$ (neither those, nor $d_\rho$ need to be known by the algorithm). The full proof of this theorem is reported in the appendix. The main steps of the proof are as follows. First we provide upper and lower confidence bounds for the estimation of the quantities $U_t(R, x, \delta)$ and $L_t(R, x, \delta)$. Then, we lower-bound the depth of the subtree of each region $(h_0, i_0) \in \mathcal{P}$ that contains a maximal point $\text{argmax}_{x \in R(h_0,i_0)} \rho(x)$, and proceed similarly for a minimal point. This uses the near-optimality dimension of $\rho$ and $-\rho$ in the region $R(h_0, i_0)$, and enables to provide an upper bound on $\hat{\boldsymbol{\rho}}_t^+(h,i;\delta)$ as well as a lower bound on $\hat{\boldsymbol{\rho}}_t^-(h,i;\delta)$. This then enables us to deduce bounds relating the estimated loss $\hat{\mathcal{L}}_t(h,i)$ to the true loss $\mathfrak{L}_{R(h,i)}(N_t)$. Finally, we relate the true loss of the current allocation to the one using the optimal one $n_{t+1}^\star(h_0, i_0)$ by discussing whether a region has been over or under sampled. This final part is closed in spirit to the proof of the regret bound for CH-AS in Carpentier et al. [2011].

In order to better understand the gain in Theorem 1, we provide the following corollary that gives more insights about the order of magnitude of the regret.

**Corollary 1** *Let $\beta \stackrel{\text{def}}{=} 1 + \ln\big(\max\{2, \gamma_2^{-d_\rho}\}\big)$. Under the assumptions of Theorem 1, assuming that the partition $\mathcal{P}$ of the space $\mathcal{X}$ is well behaved, i.e. that for all $(h_0, i_0) \in \mathcal{P}$, then $n_{t+1}^\star(h_0, i_0)$ grows at least at speed $O\Big(\ln(t)\big(\frac{1}{\gamma_2}\big)^{2h_0\beta}\Big)$, then for all $\delta \in [0,1]$, with probability higher than $1 - 2\delta$ we have*

$$\mathfrak{R}_T = O\bigg( \sum_{t=|\mathcal{P}|}^{T} \max_{(h_0, i_0) \in \mathcal{P}} \Big( \frac{1}{n_t^\star(h_0, i_0)} + 2\lambda c \gamma^{h_0} \Big) \Big( \frac{\ln(t)}{n_t^\star(h_0, i_0)} \Big)^{\frac{1}{2\beta}} \bigg).$$

This regret term has to be compared with the typical range of the cumulative loss of the optimal allocation strategy, that is given by

$$\sum_{t=|\mathcal{P}|}^{T} \mathfrak{L}_{\mathcal{P}}(n_t^\star) \;\; = \;\; \sum_{t=|\mathcal{P}|}^{T} \max_{(h_0, i_0) \in \mathcal{P}} \bigg( \frac{\rho_{(h_0, i_0)}^+}{n_t^\star(h_0, i_0)} + 2\lambda c \gamma^{h_0}(\rho_{(h_0, i_0)}^+ - \rho_{(h_0, i_0)}^-) \bigg),$$

where $\rho_{(h_0, i_0)}^+ \stackrel{\text{def}}{=} \max_{x \in R(h_0, i_0)} \rho(x)$, and similarly $\rho_{(h_0, i_0)}^- \stackrel{\text{def}}{=} \min_{x \in R(h_0, i_0)} \rho(x)$. Thus, this shows that, after normalization, the relative regret on each cell $(h_0, i_0)$ is roughly of order $\frac{1}{\rho^+(h_0, i_0)}\big(\frac{\ln(t)}{n_t^\star(h_0, i_0)}\big)^{\frac{1}{2\beta}}$, i.e. decays at speed $n_t^\star(h_0, i_0)^{-\frac{1}{2\beta}}$. This shows that we are not only able to compete with the performance of the best allocation strategy, but we actually achieve the exact same performance with multiplicative factor 1, up to a second order term. Note also that, when specified to the case of Example 1, the order of this regret is competitive with the standard results from Carpentier et al. [2011].

The lost of the variance term $\rho^+(h_0, i_0)^{-1}$ (that is actually a constant) here comes from the fact that we are only able to use Hoeffding's like bounds for the estimation of the variance. In order to remove it, one would need empirical Bernstein's bounds for variance estimation in the case of martingale difference sequences. This is postponed to future work.

## 6    Discussion

In this paper, we have provided an algorithm together with a regret analysis for a problem of online allocation of samples in a fixed partition, where the objective is to minimize a loss that contains a penalty term that is driven by a notion of curiosity. A very specific case (finite state space) already corresponds to a difficult question known as *active learning in the multi-armed bandit setting* and has been previously addressed in the literature (e.g. Antos et al. [2010], Carpentier et al. [2011]). We have considered an extension of this problem to a continuous domain where a fixed partition of the space as well as a generative model of the unknown dynamic are given, using our curiosity-driven loss function as a measure of performance. Our main result is a regret bound for that problem, that shows that our procedure is first order optimal, i.e. achieves the same performance as the best possible allocation (thus with multiplicative constant 1).

We believe this result contributes to filling the important gap that exists between existing algorithms for the challenging setting of intrinsic reinforcement learning and a theoretical analysis of such, the HORSE.C algorithm being related in spirit to, yet simpler and less ambitious the RIAC algorithm from Baranes and Oudeyer [2009]. Indeed, in order to achieve the objective that tries to address RIAC, one should first remove the assumption that the partition is given: One trivial solution is to run the HORSE.C algorithm in episodes of doubling length, starting with the trivial partition, and to select at the end of each a possibly better partition based on computed confidence intervals, however making efficient use of previous samples and avoiding a blow-up of candidate partitions happen to be a challenging question; then one should relax the generative model assumption (i.e. that we can sample wherever we want), a question that shares links with a problem called *autonomous exploration*. Thus, even if the regret analysis of the HORSE.C algorithm is already a strong, new result that is interesting independently of such difficult specific goals and of the reinforcement learning framework (no MDP structure is required), those questions are naturally left for future work.

**Acknowledgements**    The research leading to these results has received funding from the European Community's Seventh Framework Programme (FP7/2007-2013) under grant agreement no 270327 (CompLACS) and no 216886 (PASCAL2).

## Footnotes

[1]using the standard terminology in Reinforcement Learning.

[2]this assumption is only here to make calculations easier and avoid nasty technical considerations that anyway do not affect the order of the final regret bound but only concern second order terms.

# References

Andràs Antos, Varun Grover, and Csaba Szepesvàri. Active learning in heteroscedastic noise. *Theoretical Computer Science*, 411(29-30):2712–2728, 2010.

A. Baranes and P.-Y. Oudeyer. R-IAC: Robust Intrinsically Motivated Exploration and Active Learning. *IEEE Transactions on Autonomous Mental Development*, 1(3):155–169, October 2009.

Sébastien Bubeck, Rémi Munos, Gilles Stoltz, and Csaba Szepesvàri. X-armed bandits. *Journal of Machine Learning Research*, 12:1655–1695, 2011.

Alexandra Carpentier, Alessandro Lazaric, Mohammad Ghavamzadeh, Rémi Munos, and Peter Auer. Upper-confidence-bound algorithms for active learning in multi-armed bandits. In Jyrki Kivinen, Csaba Szepesvàri, Esko Ukkonen, and Thomas Zeugmann, editors, *Algorithmic Learning Theory*, volume 6925 of *Lecture Notes in Computer Science*, pages 189–203. Springer Berlin / Heidelberg, 2011.

Vincent Graziano, Tobias Glasmachers, Tom Schaul, Leo Pape, Giuseppe Cuccu, J. Leitner, and J. Schmidhuber. Artificial Curiosity for Autonomous Space Exploration. *Acta Futura (in press)*, (1), 2011.

Tobias Jung, Daniel Polani, and Peter Stone. Empowerment for continuous agent-environment systems. *Adaptive Behavior - Animals, Animats, Software Agents, Robots, Adaptive Systems*, 19(1): 16–39, 2011.

G.D. Konidaris. *Autonomous robot skill acquisition*. PhD thesis, University of Massachusetts Amherst, 2011.

Odalric-Ambrym Maillard. Hierarchical optimistic region selection driven by curiosity. *HAL*, 2012. URL http://hal.archives-ouvertes.fr/hal-00740418.

Georg Martius, J. Michael Herrmann, and Ralf Der. Guided self-organisation for autonomous robot development. In *Proceedings of the 9th European conference on Advances in artificial life*, ECAL'07, pages 766–775, Berlin, Heidelberg, 2007. Springer-Verlag.

Jonathan Mugan. *Autonomous Qualitative Learning of Distinctions and Actions in a Developing Agent*. PhD thesis, University of Texas at Austin, 2010.

Pierre-Yves Oudeyer and Frederic Kaplan. What is Intrinsic Motivation? A Typology of Computational Approaches. *Frontiers in neurorobotics*, 1(November):6, January 2007.

J. Schmidhuber. Formal theory of creativity, fun, and intrinsic motivation (1990-2010). *Autonomous Mental Development, IEEE Transactions on*, 2(3):230–247, 2010.

